# FastEx: Hash Clustering with Exponential Families

**Amr Ahmed**[*]
Research at Google, Mountain View, CA
amra@google.com

**Sujith Ravi**
Research at Google, Mountain View, CA
sravi@google.com

**Shravan M. Narayanamurthy**
Microsoft Research, Bangalore, India
shravanmn@gmail.com

**Alexander J. Smola**
Research at Google, Mountain View, CA
alex@smola.org

## Abstract

Clustering is a key component in any data analysis toolbox. Despite its importance, scalable algorithms often eschew rich statistical models in favor of simpler descriptions such as $k$-means clustering. In this paper we present a sampler, capable of estimating mixtures of exponential families. At its heart lies a novel proposal distribution using random projections to achieve high throughput in generating proposals, which is crucial for clustering models with large numbers of clusters.

## 1  Introduction

Fast clustering algorithms are a staple of exploratory data analysis. See e.g. [1] and references. Clustering is useful for partitioning data into sets of similar items. Such tools are vital e.g. in large scale document analysis, or to provide a modicum of adaptivity to content personalization for a large basis of users [2, 3]. Likewise it allows advertisers to target specific slices of the user base of an internet portal. While similarity and prototype based techniques [4, 5] satisfy a large range of these requirements, they tend to be less useful for the purpose of obtaining a proper probabilistic representation of the data. The latter, is useful for determining typical and unusual events, forecasting traffic, information retrieval, and when the results require integration into a larger *probabilistic* model.

Large scale problems, however, come with a rather surprising dilemma: as we increase the amount of data we can both estimate the model parameters for fixed model complexity (typically the number of clusters) more accurately. As a consequence we have the opportunity (and need) to increase the number of parameters, e.g. clusters. The latter is often ignored but vital to the rationale for using more data — after all, for fixed model complexity there are rapidly diminishing returns afforded by extra data once a given threshold is exceeded. See also [6, 7] for a frequentist perspective. Simply put, it is a waste of computational resources to design algorithms capable of processing big data to build a *simple* model (e.g. millions of documents for tens of clusters).

**Contributions**   We address the following problems: We need to deal with a large number of instances, e.g. by means of multicore sampling and we need to draw from a large number of clusters. When sampling from many clusters, the time to compute the object likelihood with respect to all clusters dominates the inference procedure. For instance, for 1000 clusters and documents of 1000 words a naive sampler needs to perform $10^6$ floating point operations. We can expect that a single sample will cost in excess of 1 milisecond. Given 10M documents this amounts to approximately 3 hours for a single Gibbs sampling iteration, which is clearly infeasible: sampling requires hundreds

---

[*]This work was carried out while AA, SR, SMN and AJS were with Yahoo Research.

of passes. This problem is exacerbated for hierarchical models. To alleviate this issue we use binary hashing to compute a fast proposal distribution.

## 2  Mixtures of Exponential Families

Our models are mixtures of exponential families due to their flexilibility. This is essentially an extended model of [8, 9]. For convenience we focus on mixtures of multinomials with correspondingly conjugate Dirichlet distributions. The derivations are entirely general and can be used e.g. for mixtures of Gaussians or Poisson distributions. In the following we denote by $\mathcal{X}$ the domain of observations $X = \{x_1, \dots, x_m\}$ drawn from some distribution $p$. We want to estimate $p$.

### 2.1  Exponential Families

We begin with a primer. In exponential families distributions over random variables are given by
$$p(x; \theta) = \exp\left(\langle \phi(x), \theta \rangle - g(\theta)\right). \tag{1}$$
Here $\phi : \mathcal{X} \to \mathcal{F}$ is a map from $x$ to the vector space of sufficient statistics (for simplicity assume that $\mathcal{F}$ is a Hilbert space) and $\theta \in \mathcal{F}$. Finally, $g(\theta)$ ensures that $p(x; \theta)$ is properly normalized via

$$g(\theta) := \log \int_{\mathcal{X}} \exp\left(\langle \phi(x), \theta \rangle\right) d\rho(x) \tag{2}$$

Here $\rho$ is the measure associated with $\mathcal{X}$ (e.g. the Lebesgue measure $L_2$ or a weighted counting measure for the Poisson distribution). It is well known [10] that the mean parameter associated with (1) and the maximum likelihood estimate given $X$ are connected via $\mu[\theta] = \mu[X]$ where

$$\mu[\theta] := \mathbf{E}_{x \sim p(x;\theta)}\left[\phi(x)\right] = \partial_\theta g(\theta) \text{ and } \mu[X] := \frac{1}{m} \sum_{i=1}^{m} \phi(x_i). \tag{3}$$

The mean must match the empirical average for it to be a maximum likelihood estimate.

**Example 1 (Multinomial)** *Assume that $\phi(x) = e_x \in \mathbb{R}^l$ and $\mathcal{X} = \{1, \dots, d\}$, i.e. we have a set of $d$ different outcomes and $e_x$ denotes the canonical vector associated with $x$. Empirical averages and probability estimates are directly connected via $p(x; \theta) = n_x/m = e^{\theta_x}$. Here $n_x$ denotes the number of times we observe $x$. This yields $\theta_x = \log n_x/m$ and $g(\theta) = 0$.*

### 2.2  Conjugate Priors

In general, high-dimensional maximum likelihood estimation is statistically infeasible and we require a prior on $\theta$ to obtain reliable estimates. We could impose a norm prior on $\theta$, leading to Laplace or Gaussian priors. Alternatively one may resort to conjugate priors. They have the property that the posterior distribution $p(\theta|X)$ over $\theta$ remains in the same family as $p(\theta)$ via
$$p(\theta|m_0, m_0\mu_0) = e^{\langle m_0\mu_0, \theta \rangle - m_0 g(\theta) - h(m_0, m_0\mu_0)}. \tag{4}$$
Here the conjugate prior itself is a member of the exponential family with sufficient statistic $\phi(\theta) = (\theta, -g(\theta))$ and with natural parameters $(m_0, m_0\mu_0)$. Commonly $m_0$ is referred to as concentration parameter which acts as an effective sample size and $\mu_0$ is the mean parameter describing where on the marginal polytope we expect the distribution to be. Note that $\mu_0 \in \mathcal{F}$. It corresponds to the mean of a putative distribution over observations (in a Dirichlet process this is the base measure and $m_0$ is the concentration parameter). Finally, $h(m_0, m_0\mu_0)$ is a log-partition function in the parameters of the conjugate prior. For instance, for the discrete distribution we have the Dirichlet, for the Gaussian the Gauss-Wishart, and for the Poisson distribution the Gamma. Normalization in (4) implies
$$p(\theta|X) \propto p(X|\theta)p(\theta|m_0, m_0\mu_0) \implies p(\theta|X) = p\left(\theta|m_0 + m, m_0\mu_0 + m\mu[X]\right) \tag{5}$$
We simply add the virtual observations $m_0\mu_0$ described by the conjugate prior to the actual observations $X$ and compute the maximum likelihood estimate with respect to the augmented dataset.

**Example 2 (Multinomial)** *We simply update the empirical observation counts. This yields the smoothed estimates for event probabilities in $x$:*
$$p(x; \theta) = \frac{n_x + m_0 \left[\mu_0\right]_x}{m + m_0} \text{ and equivalently } \theta_x = \log \frac{n_x + m_0 \left[\mu_0\right]_x}{m + m_0}. \tag{6}$$

## 2.3 Mixture Models

The final piece is to describe the prior over mixture components. Our tools are entirely general and could take advantage of Bayesian nonparametrics, such as the Dirichlet process or the Pitman-Yor process. For the sake of brevity and to ensure computational tractability (we need to limit the time it takes to sample from the cluster distribution for a given instance) we limit ourselves to a Dirichlet-Multinomial model with $k$ components:

- Draw discrete mixture $p(y|\theta)$ with $y \in \{1, \dots k\}$ from Dirichlet with $(m_0^{\text{cluster}}, \mu_0^{\text{cluster}})$.
- For each component $k$ draw exponential families distribution $p(x|\theta_y)$ from conjugate with parameters $(m_0^{\text{component}}, \mu_0^{\text{component}})$.
- For each $i$ first draw component $y_i \sim p(y|\theta)$, then draw observation $x_i \sim p(x|\theta_{y_i})$.

Note that we have *two* exponential families components here — a smoothed multinomial to capture cluster membership, i.e. $y \sim p(y|\theta)$ and one pertaining to the cluster distribution. Both parts *could* be joined into a single exponential family model with $y$ being the latent variable, a property that we will exploit only for the purpose of fast sampling.

The venerable EM algorithm [8] is effective for a small number of clusters. For large numbers, however, Gibbs sampling of the collapsed likelihood is computationally more advantageous since it only requires updates of the sufficient statistics of two clusters per sample, whereas EM necessitates an update of all clusters. Collapsed Gibbs sampling works as follows:

1. For a given $x_i$ draw $y_i \sim p(y_i|X, Y^{-i}) \propto p(y_i|Y) \cdot p(x_i|y_i, X^{-i}, Y^{-i})$.
2. Update the sufficient statistics for the changed clusters.

For large $k$ step 1, particularly computing $p(x_i|y_i, X^{-i}, Y^{-i})$ dominates the inference procedure. We now show how this step can be accelerated significantly using a good proposal distribution, parallel sampling, and a Taylor expansion for general exponential families.

# 3 Acceleration

## 3.1 Taylor Approximation for Collapsed Inference

Let us briefly review the key equations involved in collapsed inference. Conjugate priors allow us to integrate out the natural parameters $\theta$ and accelerate mixing in Gibbs samplers [11]. We can obtain a closed form expression for the data likelihood:

$$p(X|m_0, m_0\mu_0) = \int p(X|\theta)p(\theta|m_0, m_0\mu_0)d\theta = e^{h(m_0+m, m_0\mu_0+m\mu[X]) - h(m_0, m_0\mu_0)}. \quad (7)$$

By Bayes rule this implies that

$$p(x|X, m_0, \mu_0) \propto p(X \cup \{x\}|m_0, m_0\mu_0) \propto e^{h(m_0+m+1, m_0\mu_0+m\mu[X]+\phi(x))} \quad (8)$$

Unfortunately the normalization $h$ is often nontrivial to compute or even intractable. The exception being the multinomial, where the Laplace smoother amounts to the correct posterior $x|X$, i.e.

$$p(x|X, \mu_0, m_0) = \frac{n_x + m_0[\mu_0]_x}{m + m_0}. \quad (9)$$

In general, unfortunately, (8) will not have quite so simple form. Strictly speaking we would need to compute $h$ and perform the update directly. This can be prohibitively costly or even impossible depending on the choice of sufficient statistics. While not necessary for our running example we state the reasoning below to indicate that the problem can be overcome quite easily.

We exploit the properties of the log-partition function $h$ of the conjugate prior for an approximation:

$$\partial_{(m_0, m_0\mu_0)} h(m_0, \mu_0 m_0) = \mathbf{E}_{\theta \sim p(\theta|m_0, m_0\mu_0)} [-g(\theta), \theta] =: (-\gamma^*, \theta^*)$$

$$\text{hence } h(m_0 + 1, m_0\mu_0 + \phi(x)) \approx h(m_0, m_0\mu_0) + \langle \theta^*, \phi(x) \rangle - \gamma^*. \quad (10)$$

Here $\gamma^*$ is the expected value of the log partition function. This quantity is often hard to compute and fortunately unnecessary for inference since $\theta^*$ immediately implies a suitable normalization. Applying the Taylor expansion in $h$ to (7) yields an approximation of $x|X$ as

$$p(x|X, m_0, m_0\mu_0) \approx \exp\left(\langle\phi(x), \theta^*\rangle - g(\theta^*)\right) \qquad (11)$$

Here the normalization $g(\theta^*)$ is an immediate consequence of the fact that this is a member of the exponential family. The key advantage of (11) is that nowhere do we need to compute $h$ directly (the latter may not be available in closed form). *We only need to estimate the parameter $\theta^*$.*

**Lemma 1** *The expected parameter $\theta^* = \mathbf{E}_{\theta\sim p(\theta|X)}[\theta]$ induces at most $O(m^{-1})$ sampler error.*

PROOF. The contribution of a single instance to the sufficient statistics is $O(m^{-1})$. Since $h$ is $C_\infty$, the residual of the Taylor expansion is bounded by $O(m^{-1})$. $\square$

Hence, (11) explains why updates obtained in collapsed inference often resemble (or are identical to) a maximum-a-posteriori estimate obtained by conjugate priors, such as in Dirichlet-multinomial smoothing. The computational convenience afforded by (11) is well justified statistically.

## 3.2 Locality Sensitive Importance Sampling

The next step is to accelerate the inner product $\langle\phi(x), \theta_y^*\rangle$ in (11) since this expression is evaluated $k$ times at each Gibbs sampler step. For large $k$ this is the dominant term. We overcome this problem by using binary hashing [12]. This provides a good approximation and therefore a proposal distribution that can be used in a Metropolis-Hastings scheme without an excessive rejection rate.

To provide some motivation consider metric-based clustering algorithms such as $k$-means. They do not suffer greatly from dealing with large numbers of clusters — after all, we only need to find the closest prototype. Finding the closest point within a set in sublinear time is a well studied problem [13, 14, 15, 16]. In a nutshell it involves transforming the set of cluster centers into a data structure that is only dependent on the inherent dimensionality of the data rather than the number of objects or the dimensionality of the actual data vector.

The problem with *sampling* from the collapsed distribution is that for a proper sampler we need to consider *all* cluster probabilities *including* those related to clusters which are highly implausible and unlikely to be chosen for a given instance. That is, most of the time we discard the very computations that made sampling so expensive. This is extremely wasteful. Instead, we design a sampler which typically will only explore the clusters which are sufficiently close to the "best" matching cluster by means of a proposal distribution. [17, 12] effectively introduce binary hash functions:

**Theorem 2** *For $u, v \in \mathbb{R}^n$ and vectors $w$ drawn from a spherically symmetric distribution on $\mathbb{R}^n$ the following relation between signs of inner products and the angle $\angle(u, v)$ between vectors holds:*

$$\angle(u, v) = \pi \Pr\left\{\operatorname{sgn}\left[\langle u, w\rangle\right] \neq \operatorname{sgn}\left[\langle v, w\rangle\right]\right\} \qquad (12)$$

This follows from a simple geometric observation, namely that only whenever $w$ falls into the angle between the unit vectors in the directions of $u$ and $v$ we will have opposite signs. Any distribution of $w$ orthogonal to the plane containing $u, v$ is immaterial.

Since exponential families rely on inner products to determine the log-likelihood of how well the data fits, we can use hashing to accelerate the expensive part considerably, namely comparing data with clusters. More specifically, $\langle u, v\rangle = \|u\| \cdot \|v\| \cdot \cos\angle(u, v)$ allows us to store the signature of a vector in terms of its signs and its norm to estimate the inner product efficiently.

**Definition 3** *We denote by $h^l(v) \in \{0, 1\}^l$ a binary hash of $v$ and by $z^l(u, v)$ an estimate of the probability of matching signs, obtained as follows*

$$\left[h^l(v)\right]_i := \operatorname{sgn}\left[\langle v, w_i\rangle\right] \text{ where } w_i \sim U_m \text{ fixed and } z^l(u, v) := \frac{1}{l}\|h(u) - h(v)\|_1. \qquad (13)$$

That is, $z^l(u, v)$ measures how many bits differ between the hash vectors $h(u)$ and $h(v)$ associated with $u, v$. In this case we may estimate the unnormalized log-likelihood of an instance being assigned to a cluster via

$$s^l(x, y) = \|\theta_y\| \, \|\phi(x)\| \cos \pi z^l(\phi(x), \theta_y) - g(\theta_y) - \log n_y \tag{14}$$

We omitted the normalization $\log n$ of the cluster probability since it is identical for all components. The above can be computed efficiently for any combination of $x$ and $y$ since we can precompute (and store) the values for $\|\theta_y\|, \|\phi(x)\|, g(\theta_y), \log n_y$, and $h(\phi(x_i))$ for all observations $x_i$.

The binary representation is significant since on modern CPUs computing the Hamming distance between $h(u)$ and $h(v)$ via $z^l(u, v)$ can be achieved in a fraction of a single clock cycle by means of a vectorized instruction set. This is supported by current generation ARM and Intel CPU cores and by AMD and Nvidia GPUs (for instance Intel's SandyBridge series of processors can process up to 256 bits in one clock cycle per core) and easily accessible via compiler optimization.

### 3.3 Error Guarantees

Note, though, that $s^l(x, y)$ is not accurate, since we only use an estimate of the inner product. Hence we need to accommodate for sampling error. The following probabilistic guarantee ensures that we can turn $s^l(x, y)$ into an upper bound of the likelihood.

**Theorem 4** *Given $k \in \mathbb{N}$ mixture components and let $l$ the number of bits used for hashing. Then the unnormalized cluster log-likelihood is bounded with probability at least $1 - \delta$ by*

$$\bar{s}^l(x, y) = \|\theta_y\| \, \|\phi(x)\| \cos \left[ \pi \max \left( 0, z^l(\phi(x), \theta_y) - \sqrt{(\log k/\delta)/2l} \right) \right] - g(\theta_y) - \log n_y \tag{15}$$

PROOF. By Theorem 2 we know that in expectation the inner product can be computed via the probability of a matching sign. Since we only take a finite sample average we effectively partition this into $l$ equivalence classes. For convenience denote by $z^\infty(\phi(x), \theta_y)$ the expected value of $z^l(\phi(x), \theta_y)$ over all hash functions. By Hoeffding's theorem we know that

$$\Pr \left\{ z^\infty(\phi(x), \theta_y) < z^l(\phi(x), \theta_y) - \epsilon \right\} \leq e^{-2l\epsilon^2} \tag{16}$$

Solving for $\epsilon$ yields $\epsilon \leq \sqrt{(-\log \delta)/2l}$. Since we know that $z^\infty(\phi(x), \theta_y) \geq 0$ we can bound it for all $k$ clusters with probability $\delta$ by taking the union bound over all events with $\delta/k$ probability. $\square$

**Remark 5** *Using 128 hash bits and with a failure probability of at most $10^{-4}$ for $k = 10^4$ clusters the correction applied to $z^l(x, z)$ is less than $0.38$.*

Note that in practice we can reduce this correction factor significantly for two reasons: firstly, for small probabilities the basic Chernoff bound is considerably loose and we would be better advised to take the KL-divergence terms in the Chernoff bound directly into account, since the probability of deviation is bounded in terms of $e^{-mD(p\|p-\epsilon)}$. Secondly, we use hashing to generate a *proposal* distribution: once we select a particular cluster we verify the estimate using the true likelihood.

### 3.4 Metropolis Hastings

An alternative to using the approximate upper bound directly, we employ it as a proposal distribution in a Metropolis Hastings (MH) framework. Denote by $q$ the proposal distribution constructed from the bound on the log-likelihood after normalization. For a given $x_i$ we first sample a new cluster assignment $y_i^{\text{new}} \sim q(.)$ and then accept the proposal using (15) with probability $r$ where

$$q(y) \propto e^{\bar{s}^l(x, y)} \text{ and } r = \frac{q(y^{\text{old}})}{q(y^{\text{new}})} \frac{p(y_i^{\text{new}})p(x_i | X_{y_i^{\text{new}}}^i, m_0, \mu_0)}{p(y_i^{\text{old}})p(x_i | X_{y_i^{\text{old}}}^i, m_0, \mu_0)} \tag{17}$$

Here $p(x_i | X, m_0, \mu_0)$ is the true collapsed conditional likelihood of (8). The specific form depends on $h(.)$ as discussed in Section 3.1.

Note that for a standard collapsed Gibbs sampler, $p(x | X, \mu_0, m_0)$ would be computed for *all* $k$ candidate clusters, however, in our framework, we only need to compute it for 2 clusters: the proposed and old clusters: an $O(k)$ time saving per sample, albeit with a nontrivial rejection probability.

**Example 3** *For discrete distributions the conjugate is the Dirichlet distribution $Dir(\alpha_{1:d})$ with components given by $\alpha_j = m_0[\mu_0]_j$ and the sum of the components is given by $m_0$, where $j \in \{1 \cdots d\}$. In this case $p(x|X, \mu_0, m_0)$ reduces to predictive distribution given in (9) if $x$ is a singleton, i.e. a single observation, and to the ratio of two log partition functions if $x$ is non-singleton.[1] We have the following predictive posterior*

$$p(x_i|X, y_i, \mu_0, m_0) = \frac{\Gamma\left(\sum_{d=1}^{D} [n_d^{y_i} + \alpha_d]\right)}{\Gamma\left(\sum_{d=1}^{D} [x_d + n_d^{y_i} + \alpha_d]\right)} \prod_{d=1}^{D} \frac{\Gamma\left(x_d + n_d^{y_i} + \alpha_d\right)}{\Gamma\left(n_d^{y_i} + \alpha_d\right)}. \tag{18}$$

### 3.5 Updating the Sufficient Statistics

We conclude our discussion of past proposals by discussing the updates involved in the sufficient statistics. For the sake of brevity we focus on multinomial models. For Gaussians changes in sufficient statistics can be achieved using a low rank update of the second order matrix and its inverse. Similar operations apply to other exponential family distributions.

Whenever we assign an instance $x$ to a new cluster we need to update the sufficient statistics of the old cluster $y$ and the new cluster $y'$ via

$$(m_y - 1)\mu[X|y] \leftarrow m_y\mu[X|y] - \phi(x) \qquad\qquad m_y \leftarrow m_y - 1$$
$$(m_{y'} + 1)\mu[X|y'] \leftarrow m_{y'}\mu[X|y'] + \phi(x) \qquad\qquad m_{y'} \leftarrow m_{y'} + 1$$

Here $\mu[X|y]$ denotes the sufficient statistics for cluster $y$, i.e. it is the sufficient statistic obtained from $X$ by considering only instances for which $y_i = y$. Likewise $m_y$ the number of instances associated with $y$. This is then used to update the natural parameter $\theta_y$ and the hash representation $h(\theta_y)$. For multinomials the mean natural parameters are just log counts. Thus these updates scale as $O(W)$ where $W$ is the number of unique items (e.g. words in a document) in $x$ (for Gaussians the cost is $O(d^2)$ where $d$ is the dimensionality of the data).

The second step is to update the hash-representation. For $l$ bits a naive update would perform the dot-product between the mean natural parameters and each random vector, which scales as $O(Dl)$, where $D$ is the vocabulary size. However we can *cache* the $l$ dot product values (as floating point numbers) for each cluster and update only those dot product values. Thus if $x$ has $W$ unique words, we only incur an $O(Wl)$ penalty. Note that we never need to store the random vectors $w$ since we can compute them on the fly by means of hash functions rather than invoking a random number generator. We use `murmurhash` as a fast and high quality hash function.

## 4 Experiments

### 4.1 Data and Methods

To provide a realistic comparison on publicly available datasets we used documents from the Wikipedia collection. More specifically, we extracted the articles and category attributes from a dump of its database. We generated multiple datasets for our experiments by first sampling a set of $k$ categories and then by pooling all the articles from the chosen categories to form a document collection. This way the data was comparable and the apparent and desired diversity in terms of cluster sizes was matched. We extracted both 100 and 1000 categories, yielding the following datasets:

| | | | |
|---|---|---|---|
| $W_{100}$ | 100 clusters | 292k articles | 2.5M unique words vocabulary |
| $W_{1000}$ | 1000 clusters | 710k articles | 5.6M unique words vocabulary |

We compare our fast sampler to a more conventional uncollapsed inference procedure. That is, we compare the following two algorithms:

**Baseline** Clustering using a Dirichlet (DP) Multinomial Mixture model. It uses an uncollapsed likelihood and alternates between sampling cluster assignments and drawing from the Dirichlet distribution of the posterior.

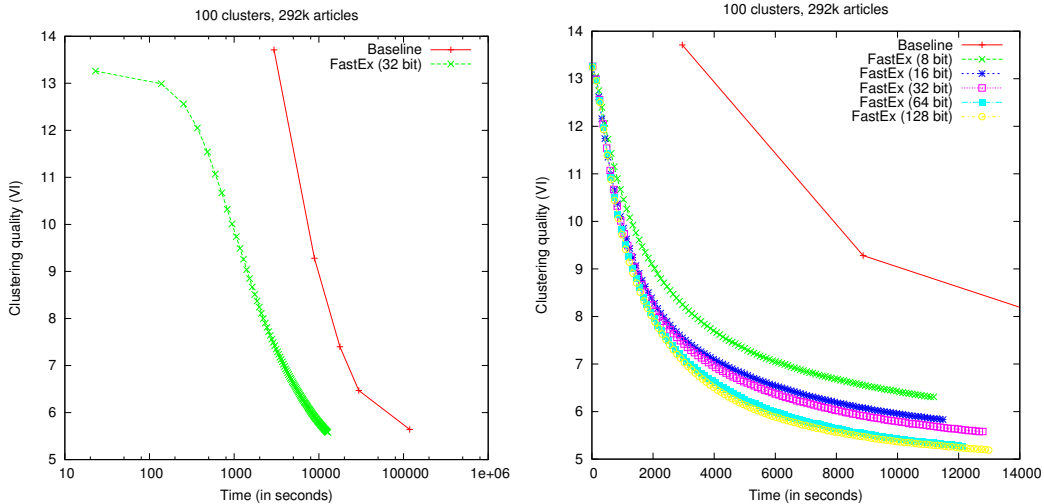

Figure 1: (Left) Convergence of both a baseline implementation and of FastEx. (Right) The effect of the hash size on performance. Note that the baseline implementation only finishes few iterations while our method almost finishes convergence.

**FastEx** We provide runtime results for a single core (our approach supports multi-core architectures, as discussed in the summary). Unless stated otherwise we use $l = 32$ bit to represent a document and cluster. This choice was made since it provides an efficient trade-off between memory usage and cost to compute the hash signatures.

## 4.2 Evaluation

For each clustering method, we report results in terms of two different measures: efficiency and clustering quality. The former is measured in terms of average run time. For the latter we use the fact that we have access to the Wikipedia category tag of each article which we treat as the gold standard for evaluation purposes.

We report results in terms of *Variation of Information* (VI) [18]. The latter is a standard measure of the distance between two clusterings. Suppose we have two clusterings (partition of a document set into several subsets) $C_1$ and $C_2$ then:

$$\text{VI}(C_1, C_2) = H(C_1) + H(C_2) - 2I(C_1, C_2) \tag{19}$$

where $H(.)$ is entropy and $I(C_1, C_2)$ is mutual information between $C_1$ and $C_2$. A lower value for VI implies a closer match to the gold standard and better quality.

We first report our results on the $W_{100}$ dataset. As shown in Figure 1 our method is an order of magnitude faster than the baseline. Hence we use a log-scale for the time axis. As evident from this Figure, our method both converges much faster than the baseline and achieves the same clustering quality. Figure 1 also displays the effect of the number of hash bits $l$ on solution quality. We vary $l \in 8, 16, \cdots, 128$ and draw the VI curve as the time goes by. As evident form the figure, increasing the number of bits caused our method to converge faster due to a tighter bound on the log-likelihood and thus a higher acceptance ratio. We also observe that beyond $64$ to $128$ bits we do not observe any noticeable improvement as predicted by our theoretical guarantees.

To see how the performance of our method changes as we increase the number of clusters, we show in Table 1 both the time required to compute the proposal distribution for a given document and the time it takes to perform the full sampling per document which includes: proposal time + time to compute acceptance ratio + time to update the clusters sufficient statistics and hash representation. As shown in this Table, thanks to the fast instruction set support for XOR and bitcount operations on modern processors, the time does not increase significantly as we increase the number of clusters and the overall time increases modestly as the number of clusters increases. Compare that to standard Collapsed Gibbs sampling in which the time scales linearly with the number of clusters.

|          | Clusters $k$ | Bitsize $l$ | **8** | **16** | **32** | **64** | **128** |
|----------|--------------|-------------|-------|--------|--------|--------|---------|
| Proposal | 100          |             | 2.34  | 2.34   | 2.34   | 2.56   | 2.90    |
| Total    |              |             | 69.52 | 69.52  | 78.77  | 81.16  | 82.19   |
| Proposal | 1000         |             | 18.80 | 18.80  | 18.80  | 21.42  | 29.12   |
| Total    |              |             | 103.91| 103.91 | 103.91 | 108.98 | 114.61  |

Table 1: Average time in microseconds spent per document for hash sampling in terms of computing the proposal distribution and total computation time. As can be seen, the total computation time for sampling 10x more clusters only increases slightly, mostly due to the increase in proposal time.

| Dataset | FastEx Quality (VI) | Baseline Quality (VI) | Speedup |
|---------|---------------------|-----------------------|---------|
| $W_{100}$ | 5.04 | 5.60 | 9.25 |
| $W_{1000}$ | 14.10 | 14.00 | 37.37 |

Table 2: Clustering quality (VI) and absolute speedup achieved by hash sampling over the baseline (DP) clustering for different *Wikipedia* datasets.

Table 2 has details on the final quality and speed up achieved by our method over the baseline. Due to a high quality proposals the time to draw from 1000 rather than 100 clusters increases slightly.

# 5   Discussion and Future Work

We presented a new efficient parallel algorithm to perform scalable clustering for exponential families. It is general and uses techniques from hashing and information retrieval to circumvent the problem of large numbers of clusters. Future work includes the application to a larger range of exponential family models and the extension of the fast retrieval scheme to hierarchical clustering.

**Parallelization**   So far we only described a single processor sampling procedure. Unfortunately this is not scalable given large amounts of data. To address the problem within single machines we use a multicore sampler to parallelize inference. This requires a small amount of approximation — rather than sampling $p(y_i|x_i, X^{-i}, Y^{-i})$ in sequence we sample up to $c$ latent variables $y_i$ in parallel in $c$ processor cores. The latter approximation is negligible since $c$ is tiny compared to the total number of documents we have. Our approach is an adaptation of the strategy described in [19]. In particular, we dissociate sampling and updating of the sufficient statistics to ensure efficient lock management and to avoid resource contention.

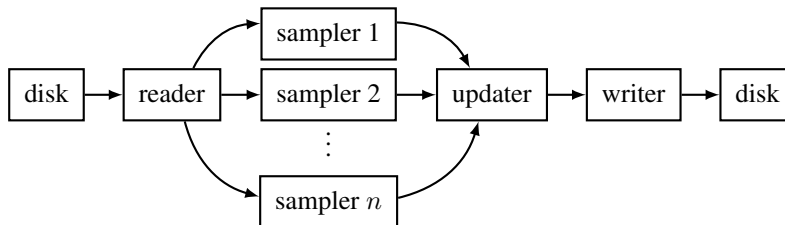

A key advantage is that all samplers share the same sufficient statistics regardless of the number of cores used. By delegating write permissions to a separate updater thread the code is considerably simplified. This allows us to be parsimonious in terms of memory use. A multi-machine setting is also achievable by keeping the sets of sufficient statistics synchronized between computers. This is possible using the synchronization architecture of [20].

**Sequential Estimation**   Our approach is compatible with sequential estimation methods and it is possible to use hash signatures for Sequential Monte Carlo estimation for clustering as in [21, 22]. However it is highly nontrivial to parallelize particle filters over a network of workstations.

**Stochastic Gradient Descent**   An alternative is to use stochastic gradient descent on a variational approximation, following the approach proposed by [23]. Again, sampling is the dominant cost for inference and it can be accelerated by binary hashing.

## Footnotes

[1] $x$ might represent a entire document $[x]_d$ denoting the count of word $d$ in $x$. The predictive distribution follows. This can be understood if we let $\phi(x) = e_x$ in the singleton case, and let $\phi(x) = ([x]_1, \cdots, [x]_D)$ in the bag-of-words case. The natural parameters of the multinomial remain the same in both cases.

# References

[1] C. D. Manning, P. Raghavan, and H. Schütze. *Introduction to Information Retrieval*. Cambridge University Press, 2008.

[2] D. Agarwal and S. Merugu. Predictive discrete latent factor models for large scale dyadic data. *Conference on Knowledge Discovery and Data Mining*, pages 26–35. ACM, 2007.

[3] A. Das, M. Datar, A. Garg, and S. Rajaram. Google news personalization: scalable online collaborative filtering. In *Conference on World Wide Web*, pages 271–280. ACM, 2007.

[4] D. Emanuel and A. Fiat. Correlation clustering — minimizing disagreements on arbitrary weighted graphs. *Algorithms — ESA 2003, 11th Annual European Symposium*, volume 2832 of *Lecture Notes in Computer Science*, pages 208–220. Springer, 2003.

[5] J. MacQueen. Some methods of classification and analysis of multivariate observations. In L. M. LeCam and J. Neyman, editors, *Proc. 5th Berkeley Symposium on Math., Stat., and Prob.*, page 281. U. California Press, Berkeley, CA, 1967.

[6] S. Negahban, P. Ravikumar, M. J. Wainwright, and B. Yu. A unified framework for high-dimensional analysis of $M$-estimators with decomposable regularizers. *CoRR*, abs/1010.2731, 2010. informal publication.

[7] V. Vapnik and A. Chervonenkis. The necessary and sufficient conditions for consistency in the empirical risk minimization method. *Pattern Recognition and Image Analysis*, 1(3):283–305, 1991.

[8] A. P. Dempster, N. M. Laird, and D. B. Rubin. Maximum Likelihood from Incomplete Data via the EM Algorithm. *Journal of the Royal Statistical Society B*, 39(1):1–22, 1977.

[9] C. E. Rasmussen. The infinite gaussian mixture model. In *Advances in Neural Information Processing Systems 12*, pages 554–560, 2000.

[10] M. J. Wainwright and M. I. Jordan. Graphical models, exponential families, and variational inference. *Foundations and Trends in Machine Learning*, $1(1 - 2):1 - 305$, 2008.

[11] T.L. Griffiths and M. Steyvers. Finding scientific topics. *Proceedings of the National Academy of Sciences*, 101:5228–5235, 2004.

[12] M. Charikar. Similarity estimation techniques from rounding algorithms. In *Proceedings of the thiry-fourth annual ACM symposium on Theory of computing*, pages 380–388, 2002.

[13] A. Beygelzimer, S. Kakade, and J. Langford. Cover trees for nearest neighbor. In *International Conference on Machine Learning*, 2006.

[14] A. Gionis, P. Indyk, and R. Motwani. Similarity search in high dimensions via hashing. In M. P. Atkinson, M. E. Orlowska, P. Valduriez, S. B. Zdonik, and M. L. Brodie, editors, *Proceedings of the 25th VLDB Conference*, pages 518–529, Edinburgh, Scotland, 1999. Morgan Kaufmann.

[15] Y. Shen, A. Ng, and M. Seeger. Fast Gaussian process regression using kd-trees. In Y. Weiss, B. Schölkopf, and J. Platt, editors, *Advances in Neural Information Processing Systems 18*, pages 1227–1234, Cambridge, MA, 2005. MIT Press.

[16] R.J. Bayardo, Y. Ma, and R. Srikant. Scaling up all pairs similarity search. In *Proceedings of the 16th international conference on World Wide Web*, pages 131–140. ACM, 2007.

[17] M.X. Goemans and D. P. Williamson. Improved approximation algorithms for maximum cut and satisfiability problems using semidefinite programming. *Journal of the ACM*, 42(6), 1995.

[18] M. Meila. Comparing clusterings by the variation of information. In *COLT*, 2003.

[19] A.J. Smola and S. Narayanamurthy. An architecture for parallel topic models. In *Very Large Databases (VLDB)*, 2010.

[20] A. Ahmed, M. Aly, J. Gonzalez, S. Narayanamurthy, and A.J. Smola. Scalable inference in latent variable models. In *Web Science and Data Mining (WSDM)*, 2012.

[21] A. Ahmed, Q. Ho, C. H. Teo, J. Eisenstein, A. J. Smola, and E. P. Xing. Online inference for the infinite cluster-topic model: Storylines from streaming text. In *AISTATS*, 2011.

[22] A. Ahmed, Q. Ho, J. Eisenstein, E. P. Xing, A. J. Smola, and C. H. Teo. Unified analysis of streaming news. In *www*, 2011.

[23] D. Mimno, M. Hoffman, and D. Blei. Sparse stochastic inference for latent dirichlet allocation. In *International Conference on Machine Learning*, 2012.

